# 3 state neurons for contextual processing

**Adam Kepecs\* and Sridhar Raghavachari**
Volen Center for Complex Systems
Brandeis University
Waltham MA 02454
*{kepecs,sraghava}@brandeis.edu*

## Abstract

Neurons receive excitatory inputs via both fast AMPA and slow NMDA type receptors. We find that neurons receiving input via NMDA receptors can have two stable membrane states which are input dependent. Action potentials can only be initiated from the higher voltage state. Similar observations have been made in several brain areas which might be explained by our model. The interactions between the two kinds of inputs lead us to suggest that some neurons may operate in 3 states: disabled, enabled and firing. Such enabled, but non-firing modes can be used to introduce context-dependent processing in neural networks. We provide a simple example and discuss possible implications for neuronal processing and response variability.

## 1 Introduction

Excitatory interactions between neurons are mediated by two classes of synapses: AMPA and NMDA. AMPA synapses act on a fast time scale ($\tau_{AMPA} \sim$ 5ms), and their role in shaping network dynamics has been extensively studied. The NMDA type receptors are slow (($\tau_{NMDA} \sim$ 150ms) and have been mostly investigated for their critical role in the induction of long term potentiation, which is thought to be the mechanism for storing long term memories. Crucial to this is the unique voltage dependence of NMDA receptors [6] that requires both the presynaptic neuron to be active and the post-synaptic neuron to be depolarized for the channel to open. However, pharamacological studies which block the NMDA receptors impair a variety of brain processes, suggesting that NMDA receptors also play a role in shaping the dynamic activity of neural networks [10, 3, 8, 11, 2].

Therefore, we wanted to examine the role of NMDA receptors in post-synaptic integration. Harsch and Robinson [4] have observed that injection of NMDA conductance that simulates synchronous synaptic input regularized firing while lowering response reliability. Our initial observations using a minimal model with

---

large NMDA inputs in a leaky dendrite showed a large regenerative depolarization. Neurons however, also possess a variety of potassium currents that are able to limit these large excursions in voltage. In particular, recent observations show that A-type potassium currents are abundant in dendrites of a variety of neurons [7]. Combining these potassium currents with random NMDA inputs showed that the membrane voltage alternated between two distinct *subthreshold* states. Similar observations of two-state fluctuations have been made *in vivo* in several cortical areas and the striatum [17, 9, 1]. The origin and possible functional relevance of these fluctuations have remained a puzzle. We suggest that the NMDA type inputs combined with potassium currents are sufficient to produce such membrane dynamics. Our results lead us to suggest that the fluctuations could be used to represent contextual modulation of neuronal firing.

## 2 NMDA-type input causes 2 state membrane fluctuations

### 2.1 Model

To examine the role of NMDA type inputs, we built a simple model of a cortical neuron receiving AMPA and NMDA type inputs. To capture the spatial extent of neuronal morphology we use a two-compartment model of pyramidal neurons [15]. We represent the soma, proximal dendrites and the axon lumped into one compartment containing the channels necessary for spike generation ($I_{Na}$ and $I_K$). The dendritic compartment includes two potassium currents, a fast activating $I_{KA}$ and the slower $I_{KS}$ along with a persistent sodium current $I_{NaP}$. The dendrite also receives synaptic input as $I_{NMDA}$ and $I_{AMPA}$.

The membrane voltage of the neuron obeys the current balance equations:

$$C_m \frac{dV_i^{\mathrm{s}}}{dt} = -I_{Leak,i}^{\mathrm{s}} - I_{Na,i} - I_{KDr,i} - \frac{g_c}{p}(V_i^{\mathrm{s}} - V_i^{\mathrm{d}}) - I_{syn}, \tag{1}$$

while the dendritic voltage, $V_i^{\mathrm{d}}$ obeys:

$$C_m \frac{dV_i^{\mathrm{d}}}{dt} = -I_{Leak,i}^{\mathrm{d}} - I_{NaP,i} - I_{Ks,i} - I_{KA,i} - \frac{g_c}{1-p}(V_i^{\mathrm{d}} - V_i^{\mathrm{s}}) - I_{syn}, \tag{2}$$

where $C_m$ is the specific membrane capacitance which is taken to be 1 $\mu$F/cm$^2$ for both the dendrite and the soma for all cells and $p =$0.2, $g_c =$0.05 determining the electrotonic structure of the neuron.

The passive leak current in both the soma and dendrites were modeled as $I_{Leak} = g_{leak}(V - E_{leak})$, where $g_{leak}$ was the leak conductance which was taken to be 0.3 mS/cm$^2$ for the soma and dendrite. $E_{leak} =$ -80mV was the leak reversal potential for both the compartments. The voltage-dependent currents were modeled according to the Hodgkin-Huxley formalism, with the gating variables obeying the equation:

$$\frac{dx}{dt} = \phi_x(\alpha_x(V)(1-x) - \beta_x(V)x) = \phi_x\left(\frac{x_\infty(V) - x}{\tau_x(V)}\right), \tag{3}$$

where $x$ represents the activation/inactivation gates for the voltage-dependent currents.

The sodium current, $I_{Na} = g_{Na}m_\infty^3 h(V^s - E_{Na})$, where $g_{Na} = 45$ mS/cm$^2$ and sodium reversal potential, $E_{Na} = 55$ mV with $m_\infty(V) = \frac{\alpha_m(V)}{\alpha_m(V)+\beta_m(V)}$. The activation variables, $\alpha_m(V) = -0.1(V + 32)/[\exp(-(V + 32)/10) - 1]$, $\beta_m(V) = 4\exp(-[V + 57]/18)$; $\alpha_h(V) = 0.07\exp(-[V + 48]/20)$ and $\beta_h(V) = 1/[\exp(-\{V + 18\}/10) + 1]$, with $\phi_m = \phi_h = 2.5$.

The delayed rectifier potassium current, $I_{KDr} = g_K n^4(V^s - E_K)$, where $g_K = 9$ mS/cm$^2$ and potassium reversal potential, $E_K = -80$ mV with $\alpha_n(V) = -0.01(V + 34)/[\exp(-(V + 34)/10) - 1]$, $\beta_n(V) = 0.125\exp(-[V + 44]/80)$, with $\phi_n = 2.5$.

In the dendrite, the persistent sodium current, $I_{NaP} = g_{NaP}r_\infty^3(V)(V - V_{Na})$, with $r_\infty(V) = 1/(1 + \exp(-(V + 57)/5))$ and $g_{NaP} = 0.25$ mS/cm$^2$. The two potassium currents were $I_{KS} = g_{Ks}q(V - V_K)$, with $q_\infty(V) = 1/(1 + \exp(-(V + 50)/2))$ and $\tau_q(V) = 200/(\exp(-(V + 60)/10) + \exp((V + 60)/10))$ and $g_{KS} = 0.1$ mS/cm$^2$; and $I_{KA} = g_{KA}a_\infty^3(V)b(V - V_K)$, with $a_\infty(V) = 1/(1 + \exp(-(V + 45)/6))$, $b_\infty(V) = 1/(1 + \exp(-(V + 56)/15))$ and $\tau_b(V) = 2.5(1 + \exp((V + 60)/30))$ and $g_{KA} = 10$ mS/cm$^2$.

The NMDA current, $I_{\text{NMDA}} = fg_{\text{NMDA}}s(V - E_{\text{NMDA}})/(1 + 0.3[Mg]\exp(-0.08V))$, where $s$ was the activation variable and $f$ denoted the inactivation of NMDA channels due to calcium entry. AMPA and NMDA inputs were modeled as conductance kicks that decayed with $\tau_{AMPA} = 5$ ms and $\tau_{\text{NMDA}} = 150$ ms. Calcium dependent inactivation of the NMDA conductance was modeled as a negative feedback $df/dt = (f_\infty - f)/2$, where $f_\infty$ was a shallow sigmoid function that was 1 below a conductance threshold of 2 ms/cm$^2$ and was inversely proportional to the NMDA conductance above threshold. The coupling conductance is $g_c = 0.1$ mS/cm $^2$. The asymmetry between the areas of the two compartments is taken into account in the parameter $p = somatic\ area/total\ area = 0.2$. The temperature scaling factors are $\phi_h = \phi_n = 3.33$. Other parameter values are: $g_{\text{Leak}} = 0.3$, $g_{\text{Na}} = 36$, $g_K = 6$, $g_{\text{NaP}} = 0.15$, $g_{KS} = 1$, $g_{KA} = 50$ in mS/cm$^2$ unless otherwise noted; $E_{\text{Leak}} = -75$, $E_{\text{Na}} = +55$, $E_K = -90$, $E_{KA} = -80$ in mV. Synchronous inputs were modeled as a compound Poisson process representing 100 inputs firing at a rate $\lambda$ each spiking with a probability of 0.1. Numerical integration was performed with a fourth-order Runge-Kutta method using a 0.01 ms time step.

## 2.2  NMDA induced two-state fluctuations

Figure 1A shows the firing produced by inputs with high AMPA/NMDA ratio. Figure 1B shows that the same spike train input delivered via synapses with a high NMDA content results in robust two-state membrane behavior. We term the lower and higher voltage states as *UP* and *DOWN* states respectively. Spikes caused by AMPA-type inputs only occur during the up-state. In general, the same AMPA input can only elicit spikes in the postsynaptic neuron when the NMDA input switches that neuron into the up-state.

Transitions from down to up-state occur when synchronous NMDA inputs depolarize the membrane enough to cause the opening of additional NMDA receptor channels (due to the voltage-dependence of their opening). This results in a regeneretive depolarization event, which is limited by the fast opening of $I_{KA}$-type

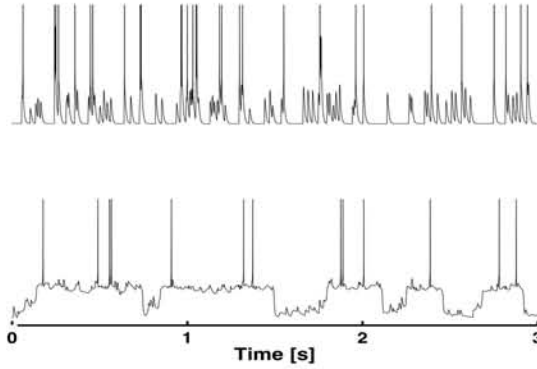

Figure 1: Inputs with high AMPA-NMDA ratio cause the cell to spike (top trace, $g_{\text{AMPA}}$ =0.05, $g_{\text{NMDA}}$ =0.01). Strong NMDA inputs combined with potassium currents (for the same AMPA input) result in fluctuations of the membrane potential between two subthreshold states, with occasional firing due to the AMPA inputs (bottom trace, $g_{\text{AMPA}}$ =0.01, $g_{\text{NMDA}}$ =0.1)

potassium channels. This up-state is stable because the regenerative nature and long lifetime of NMDA receptor opening keeps the membrane depolarized, while the slower $I_{KS}$ potassium current prevents further depolarization. When input ceases, NMDA channels eventually ($\tau_{NMDA} \sim 150ms$) close and the membrane jumps to the down-state. Note that while this bistable mechanism *is* intrinsic to the membrane, it is also conditional upon input. Since the voltage threshold for spike generation in the soma/axon compartment is above the up-state, it acts as a barrier. Thus, synchronous AMPA input in the down-state has a low probability of eliciting a spike.

A number of previous experimental studies have reported similar phenomena in various brain regions [16, 9, 1] where the two states persist even with all intrinsic inward currents blocked but the inputs left intact [17]. Pharmacological block of the potassium currents resulted in prolonged up-states [17]. These experimental results suggested a conceptual model in which two-state fluctuations are (i) input driven, (ii) the membrane states are stabilized by potassium currents. Nevertheless, there remained a puzzle that (iii) up-state transitions are abrupt and (iv) the the up-state is prolonged and restricted to a relatively narrow range of voltages. Our model suggests a plausible mechanism for this phenomenon consistent with all experimental constraints. Below, we examine the origins of the two-state fluctuations in light of these findings.

### 2.3 Analysis of two state fluctuations

Figure 2A shows the histogram of membrane potential for a neuron driven by combined AMPA and NMDA input at 30 Hz. There are two clear modes corresponding to the up and down-states. The variability of the up-state and down-state voltages is very low ($\sigma = 1.4$ mV and 2.4 mV respectively) as observed. Figure 2B shows the distribution of the up-state duration. The distribution of the up-state durations depend on the maximal NMDA conductance and the decay time constant of NMDA (not shown), as well as the mean rate of NMDA inputs (Figure 2C).

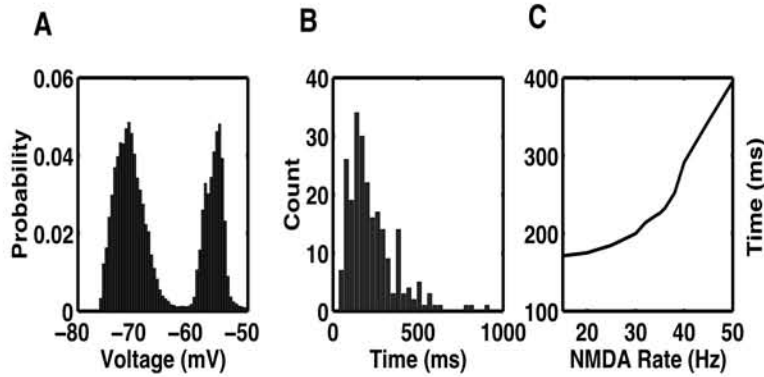

Figure 2: A. Histogram of the up and down states. B. Dwell times of the up states C. Mean duration of the up states increases with rate of NMDA inputs. Each histogram was calculated over a run of 120 seconds.

Additionally, larger maximal potassium conductances shorten the duration of the up states. Thus, we predict that the NMDA receptors are intimately involved in shaping the firing characteristics of these neurons. Furthermore, our mechanistic explanation leads a strong prediction about the functional role for these fluctuations in neuronal processing.

## 3 Contextual processing with NMDA and AMPA pathways

Since NMDA and AMPA pathways have distinct roles in respectively switching and firing our model neuron, we suggest the following conceptual model shown on Fig 3A. Without any input the neuron is at the rest or *disabled state*. Contextual input (via NMDA receptors) can bring the neuron into an *enabled state*. Informational (for instance, cue or positional) input (via AMPA receptors) can fire a neuron only from this enabled state.

Where might such an architecture be used? In the CA1 region of the hippocampus, pyramidal cells receive two distinct, spatially segregated input pathways: the perforant path from cortex and the Schaffer collaterals from the CA3 region. The perforant path has a very large NMDA receptor content [14] which is, interestingly, co-localized with high densities of $I_{KA}$ conductances [5]. Experimental [13] and theoretical [12] observations suggest that these two pathways carry distinct information. Lisman has suggested that the perforant path carries contextual information and the Schaffer collaterals bring sequence information [12]. Thus our model seems to apply biophysically as well as suggest a possible way for CA1 neurons to carry out contextual computations. It is known that these cell can fire at specific places in specific contexts. How might these different signals interact? As shown on Fig3B, our model neuron can only fire spikes due to positional input when the right context enables it. We note that a requirement for contextual processing is that the two inputs be anatomically segregated, as they are in the CA1 region. However, we stress that the phenomenon of 2-state fluctuations itself is independent of the location of the two kinds of inputs.

Figure 4A shows a similar processing scheme adapted for higher-order language

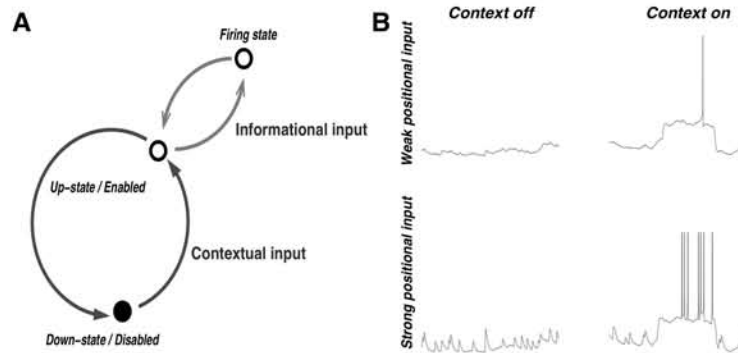

Figure 3: A. Contextual input (high NMDA) switches the neuron from a rest state to an up state. Informational input (high AMPA) cause the neuron to spike only from the up state. B. Weak informational input can cause the cell to fire in conjunction with the contextual input, (left traces) while strong informational input will not fire the cell in the absence of contextual input (right traces). In this simulation, the soma/proximal dendrite compartment receives AMPA input, while the NMDA input targets the dendritic compartment.

processing. We simulated 3 neurons each receiving the same AMPA, informational input. This might represent the word "green". Each of these neurons also receives distinct contextual input via NMDA type receptors. These might, for instance, represent specific noun groups: objects, people and fruit. The word "green" may have very different meanings in these different contexts such as the color green, a person who is a novice or an unripe fruit. We simulated this simple scenario shown in Figure 4C. Even though each neuron receives the same strong AMPA input, their firing seems uncorrelated. To evaluate the performance of the network in processing contextual conjunctions, we measured the correlations between the information and each contextual input. The most correlated at each moment was designated to be the correct meaning. We then measured the number of spikes emitted by each neuron during each "meaning". Figure 4B shows that the neurons performed well, each tuned to fire preferentially in its appropriate context.

This simple example illustrates the use of a plausible biophysical mechanism for computing conjuctions or multiplying with neurons.

## 4    Discussion

Voltage fluctuations between two subthresold levels with similar properties are observed *in vivo* in a variety of brain regions. Our model is in accordance with these data and lead us to a new picture of how might these neuron operate in a functional manner. Figure 3A shows our model operating as a 3-state device. It has a stable low membrane state from which it cannot fire spikes, which we called disabled. It also has a stable depolarized state from which action potentials can be elicited, which this we call enabled state. Additionally, it has a firing state which is only reachable from the enabled state.

What might be the role of the two non-firing states? We suggest that if high and low NMDA-content pathways carry separate information these neurons can compute

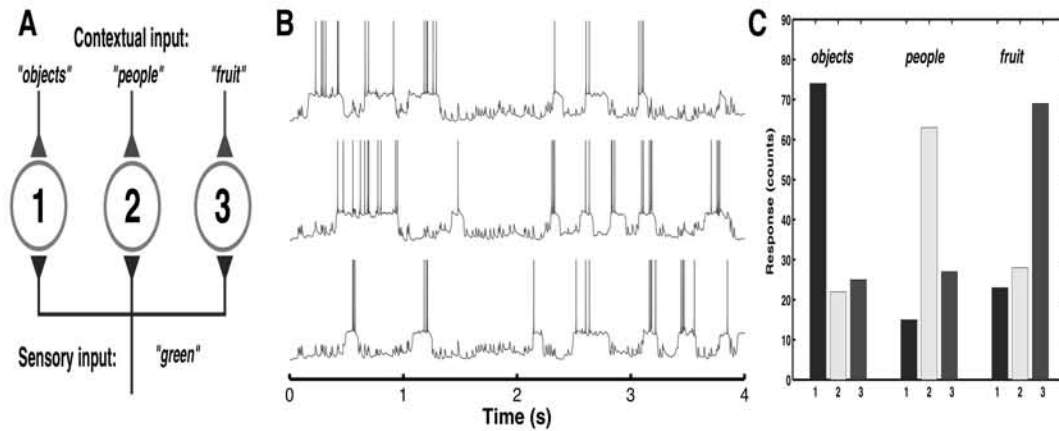

Figure 4: A. Illustrative task for contextual processing in semantic inference. 3 neurons each receive independent contextual (NMDA) and common informational (AMPA) input. B. Voltage traces showing differences in firing patterns depending upon context. C. Each neuron is tuned to its defined context. Correlation was measured between the informational spike train and each contextual spike train smoothed with a gaussian filter ($\sigma = 60ms$). The most correlated context was defined to be the right one and the spikes of all neurons were counted.

conjuctions, a simple form of multiplication. If the high NMDA-content pathway carries contextual information then it would be in position to enable or disable a neuron. In the enabled state, AMPA-type informational input could then fire a neuron (Fig 3B).

We have presented a biophysical model for two-state fluctuations that is strongly supported by data. One concern might be that most observations of 2-state fluctuations *in vivo* have been when the animal is anesthetized, implying that this kind of neuronal dynamics is an artifact of the anesthetized state. However, these fluctuations have been observed in several different kinds of anesthesia, including local anesthesia [16]. Furthermore, it has been shown that the duration of the up-states correlate with orientation selectivity in visual cortical neurons suggesting that these fluctuations might play a role in information processing. These observations suggest that this phenomenon may be more indicative of a natural state of the cortex rather than a by-product of anesthesia.

When the inputs with different AMPA/NMDA content are anatomically segregated, the NMDA input alone generates voltage fluctuations between a resting and depolarized state, while the AMPA input causes the neuron to spike when in the up-state. This mechanism naturally leads to the suggestion that such two-state fluctuations could have a function in computing context/input conjuctions. In summary, we suggest the known biophysical mechanisms of some neurons can enable them two operate as 3-state devices. In this mode of operation, the neurons could be used for contextual processing.

## Acknowledgments

We acknowledge John Lisman and John Fitzpatrick for useful discussion and suggestions.

# References

[1] J. Anderson, I. Lampl, I. Reichova, M. Carandini, and D. Ferster. Stimulus dependence of two-state fluctuations of membrane potential in cat visual cortex. *Nat Neurosci*, 3:617–21, 2000.

[2] A. Compte, N. Brunel, P. Goldman-Rakic, and X.J. Wang. Synaptic mechanisms and network dynamics underlying spatial working memory in a cortical network model. *Cereb. Cortex*, 10(9):910–923, 2000.

[3] S. Grillner, O. Ekeberg, A. Manira, A. Lansner, D. Parker, J. Tegner, and P. Wallen. Intrinsic function of a neuronal network – a vertebrate central pattern generator. *Brain Res. Brain Res. Rev.*, 26:184–197, 1998.

[4] A. Harsch and H.P.C. Robinson. Postsynaptic variability of firing rates in rat cortical neurons: the role of input synchronization and synaptic nmda receptor conductance. *J. Neurosci.*, 20:6181–6192, 2000.

[5] A Hoffman, JC Magee, CM Colbert, and D Johnston. $K^+$ channel regulation of signal propagation in dendrites of hippocampal pyramidal neurons. *Nature*, 387:869–875, 1997.

[6] C.E. Jahr and C.F. Stevens. Voltage dependence of nmda-activated macroscopic conductances predicted by single-channel kinetics. *J Neurosci*, 10:3178–82, 1990.

[7] D. Johnston, D.A. Hoffman, J.C. Magee, N.P. Poolos, S. Watanabe, C.M. Colbert, and M. Migliore. Dendritic potassium channels in hippocampal pyramidal neurons. *J Physiol*, 15:75–81, 2000.

[8] O. Kiehn and T. Eken. Functional role of plateau potentials in vertebrate motor neurons. *Curr. Opin. Neurobiol.*, 8:746–752, 1998.

[9] B.L. Lewis and P. O'Donnell. Ventral tegmental area afferents to the prefrontal cortex maintain membrane potential 'up' states in pyramidal neurons via $d_1$ dopamine receptors. *Cereb. Cortex*, 10:1168–1175, 2000.

[10] Y.X. Li, R. Bertram, and J. Rinzel. Modeling N-methyl-D-aspartate induced bursting in dopamine neurons. *Neuroscience*, 71(2):397–410, 1996.

[11] J. Lisman, J.-M. Fellous, and X.J. Wang. A role for NMDA-receptor channels in working memory. *Nat. Neurosci.*, 1(4):273–275, 1998.

[12] J.E. Lisman. Relating hippocampal circuitry to function: recall of memory sequences by reciprocal dentate-CA3 interactions. *Neuron*, 22:233–242, 1999.

[13] B.L. McNaughton, C.A. Barnes, J. Meltzer, and R.J. Sutherland. Hippocampal granule cells are necessry for normal spatial learning but not for spatially-selective pyramidal cell discharge. *Exp. Brain Res.*, 76:485–496, 1989.

[14] N.A. Otmakhova and Lisman J. Dopamine selectively inhibits the direct cortical pathway to the CA1 hippocampal region. *J Neurosci*, 19:1437–45, 1999.

[15] P.F. Pinsky and J. Rinzel. Intrinsic and network rhythmogenesis in a reduced Traub model for CA3 neurons. *J. Comput. Neurosci.*, 1:39–60, 1994.

[16] C.J. Wilson and P.M. Groves. Spontaneous firing patterns of identified spiny neurons in the rat neostriatum. *Brain Res*, 220:67–80, 1981.

[17] C.J. Wilson and Y. Kawaguchi. The origins of two-state spontaneuous fluctuations of neostriatal spiny neurons. *J. Neurosci.*, 16:2397–2410, 1996.
